# Ocular Dominance and Patterned Lateral Connections in a Self-Organizing Model of the Primary Visual Cortex

**Joseph Sirosh and Risto Miikkulainen**
Department of Computer Sciences
University of Texas at Austin, Austin, TX 78712
email: sirosh,risto@cs.utexas.edu

## Abstract

A neural network model for the self-organization of ocular dominance and lateral connections from binocular input is presented. The self-organizing process results in a network where (1) afferent weights of each neuron organize into smooth hill-shaped receptive fields primarily on one of the retinas, (2) neurons with common eye preference form connected, intertwined patches, and (3) lateral connections primarily link regions of the same eye preference. Similar self-organization of cortical structures has been observed experimentally in strabismic kittens. The model shows how patterned lateral connections in the cortex may develop based on correlated activity and explains why lateral connection patterns follow receptive field properties such as ocular dominance.

## 1 Introduction

Lateral connections in the primary visual cortex have a patterned structure that closely matches the response properties of cortical cells (Gilbert and Wiesel 1989; Malach et al. 1993). For example, in the normal visual cortex, long-range lateral connections link areas with similar orientation preference (Gilbert and Wiesel 1989). Like cortical response properties, the connectivity pattern is highly plastic in early development and can be altered by experience (Katz and Callaway 1992). In a cat that is brought up squint-eyed from birth, the lateral connections link areas with the same ocular dominance instead of orientation (Löwel and Singer 1992). Such patterned lateral connections develop at the same time as the orientation selectivity and ocular dominance itself (Burkhalter et al. 1993; Katz and Callaway 1992). Together,

these observations suggest that the same experience-dependent process drives the development of both cortical response properties and lateral connectivity.

Several computational models have been built to demonstrate how orientation preference, ocular dominance, and retinotopy can emerge from simple self-organizing processes (e.g. Goodhill 1993; Miller 1994; Obermayer et al. 1992; von der Malsburg 1973). These models assume that the neuronal response properties are primarily determined by the afferent connections, and concentrate only on the self-organization of the afferent synapses to the cortex. Lateral interactions between neurons are abstracted into simple mathematical functions (e.g. Gaussians) and assumed to be uniform throughout the network; lateral connectivity is not explicitly taken into account. Such models do not explicitly replicate the activity dynamics of the visual cortex, and therefore can make only limited predictions about cortical function.

We have previously shown how Kohonen's self-organizing feature maps (Kohonen 1982) can be generalized to include self-organizing lateral connections and recurrent activity dynamics (the Laterally Interconnected Synergetically Self-Organizing Map (LISSOM); Sirosh and Miikkulainen 1993, 1994a), and how the algorithm can model the development of ocular dominance columns and patterned lateral connectivity with abstractions of visual input. LISSOM is a low-dimensional abstraction of cortical self-organizing processes and models a small region of the cortex where all neurons receive the same input vector. This paper shows how realistic, high-dimensional receptive fields develop as part of the self-organization, and scales up the LISSOM approach to large areas of the cortex where different parts of the cortical network receive inputs from different parts of the receptor surface. The new model shows how (1) afferent receptive fields and ocular dominance columns develop from simple retinal images, (2) input correlations affect the wavelength of the ocular dominance columns and (3) lateral connections self-organize cooperatively and simultaneously with ocular dominance properties. The model suggests new computational roles for lateral connections in the cortex, and suggests that the visual cortex maybe maintained in a continuously adapting equilibrium with the visual input by coadapting lateral and afferent connections.

## 2   The LISSOM Model of Receptive Fields and Ocular Dominance

The LISSOM network is a sheet of interconnected neurons (figure 1). Through afferent connections, each neuron receives input from two "retinas". In addition, each neuron has reciprocal excitatory and inhibitory lateral connections with other neurons. Lateral excitatory connections are short-range, connecting only close neighbors. Lateral inhibitory connections run for long distances, and may even implement full connectivity between neurons in the network.

Neurons receive afferent connections from broad overlapping patches on the retina called anatomical receptive fields, or RFs. The $N \times N$ network is projected on to each retina of $R \times R$ receptors, and each neuron is connected to receptors in a square area of side $s$ around the projections. Thus, neurons receive afferents from corresponding regions of each retina. Depending on the location of the projection, the number of afferents to a neuron from each retina could vary from $\frac{1}{2}s \times \frac{1}{2}s$ (at the corners) to $s \times s$ (at the center).

The external and lateral weights are organized through an unsupervised learning process. At each training step, neurons start out with zero activity. The initial response $\eta_{ij}$ of neuron $(i, j)$

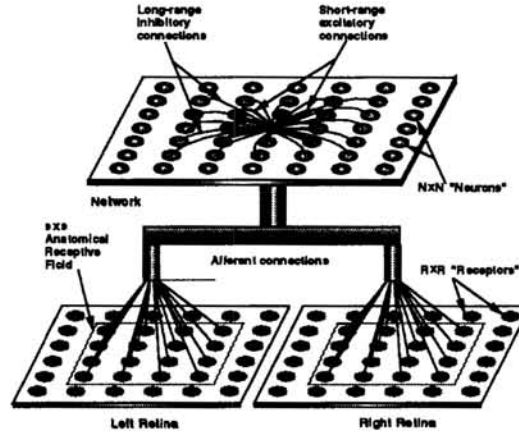

Figure 1: **The Receptive-Field LISSOM architecture.** The afferent and lateral connections of a single neuron in the LISSOM network are shown. All connection weights are positive.

is based on the scalar product

$$\eta_{ij} = \sigma \left( \sum_{a,b} \xi_{ab} \mu_{ij,ab} + \sum_{c,d} \xi_{cd} \mu_{ij,cd} \right), \tag{1}$$

where $\xi_{ab}$ and $\xi_{cd}$ are the activations of retinal receptors $(a, b)$ and $(c, d)$ within the receptive fields of the neuron in each retina, $\mu_{ij,ab}$ and $\mu_{ij,cd}$ are the corresponding afferent weights, and $\sigma$ is a piecewise linear approximation of the familiar sigmoid activation function. The response evolves over time through lateral interaction. At each time step, the neuron combines the above afferent activation $\sum \xi \mu$ with lateral excitation and inhibition:

$$\eta_{ij}(t) = \sigma \left( \sum \xi \mu + \gamma_e \sum_{k,l} E_{ij,kl} \eta_{kl}(t-1) - \gamma_i \sum_{k,l} I_{ij,kl} \eta_{kl}(t-1) \right), \tag{2}$$

where $E_{ij,kl}$ is the excitatory lateral connection weight on the connection from neuron $(k, l)$ to neuron $(i, j)$, $I_{ij,kl}$ is the inhibitory connection weight, and $\eta_{kl}(t-1)$ is the activity of neuron $(k, l)$ during the previous time step. The constants $\gamma_e$ and $\gamma_i$ determine the relative strengths of excitatory and inhibitory lateral interactions. The activity pattern starts out diffuse and spread over a substantial part of the map, and converges iteratively into stable focused patches of activity, or activity bubbles. After the activity has settled, typically in a few iterations of equation 2, the connection weights of each neuron are modified. Both afferent and lateral weights adapt according to the same mechanism: the Hebb rule, normalized so that the sum of the weights is constant:

$$w_{ij,mn}(t + \delta t) = \frac{w_{ij,mn}(t) + \alpha \eta_{ij} X_{mn}}{\sum_{mn} [w_{ij,mn}(t) + \alpha \eta_{ij} X_{mn}]}, \tag{3}$$

where $\eta_{ij}$ stands for the activity of neuron $(i, j)$ in the final activity bubble, $w_{ij,mn}$ is the afferent or lateral connection weight ($\mu$, $E$ or $I$), $\alpha$ is the learning rate for each type of connection ($\alpha_a$ for afferent weights, $\alpha_E$ for excitatory, and $\alpha_I$ for inhibitory) and $X_{mn}$ is the presynaptic activity ($\xi$ for afferent, $\eta$ for lateral).

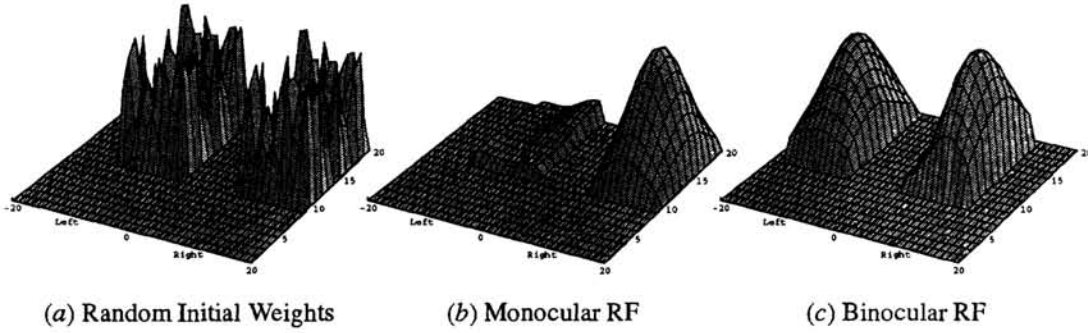

(a) Random Initial Weights      (b) Monocular RF      (c) Binocular RF

Figure 2: **Self-organization of the afferent input weights into receptive fields.** The afferent weights of a neuron at position $(42, 39)$ in a $60 \times 60$ network are shown before (a) and after self-organization (b). This particular neuron becomes monocular with strong connections to the right eye, and weak connections to the left. A neuron at position $(38, 23)$ becomes binocular with appoximately equal weights to both eyes (c).

Both excitatory and inhibitory lateral connections follow the same Hebbian learning process and strengthen by correlated activity. The short-range excitation keeps the activity of neighboring neurons correlated, and as self-organization progresses, excitation and inhibition strengthen in the vicinity of each neuron. At longer distances, very few neurons have correlated activity and therefore most long-range connections become weak. Such weak connections are eliminated, and through weight normalization, inhibition concentrates in a closer neighborhood of each neuron. As a result, activity bubbles become more focused and local, weights change in smaller neighborhoods, and receptive fields become better tuned to local areas of each retina.

The input to the model consists of gaussian spots of "light" on each retina:

$$\xi_{x,y} = exp(-\frac{(x - x_i)^2 + (y - y_i)^2}{u^2}) \qquad (4)$$

where $\xi_{x,y}$ is the activation of receptor $(x, y)$, $u^2$ is a constant determining the width of the spot, and $(x_i, y_i)$: $0 \le x_i, y_i < R$ its center. At each input presentation, one spot is randomly placed at $(x_i, y_i)$ in the left retina, and a second spot within a radius of $\rho \times RN$ of $(x_i, y_i)$ in the right retina. The parameter $\rho \in [0, 1]$ specifies the spatial correlations between spots in the two retinas, and can be adjusted to simulate different degrees of correlations between images in the two eyes.

## 3   Simulation results

To see how correlation between the input from the two eyes affects the columnar structures that develop, several simulations were run with different values of $\rho$. The afferent weights of all neurons were initially random (as shown in figure 2a), with the total strength to both eyes being equal.

Figures 2b,c show the final afferent receptive fields of two typical neurons in a simulation with $\rho = 1$. In this case, the inputs were uncorrelated, simulating perfect strabismus. In the early stages of such simulation, some of the neurons randomly develop a preference for one eye or the other. Nearby neurons will tend to share the same preference because lateral

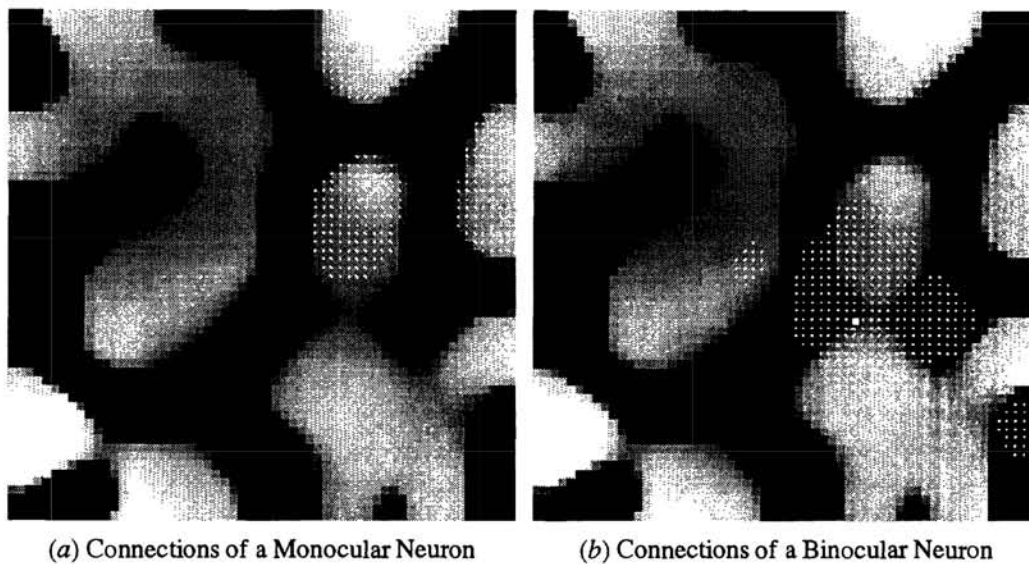

(*a*) Connections of a Monocular Neuron      (*b*) Connections of a Binocular Neuron

Figure 3: **Ocular dominance and lateral connection patterns.** The ocular dominance of a neuron is measured as the difference in total afferent synaptic weight from each eye to the neuron. Each neuron is labeled with a grey-scale value (*black* → *white*) that represents continuously changing eye preference from exclusive left through binocular to exclusive right. Small white dots indicate the lateral input connections to the neuron marked with a big white dot. (*a*) The surviving lateral connections of a left monocular neuron predominantly link areas of the same ocular dominance. (*b*) The lateral connections of a binocular neuron come from both eye regions.

excitation keeps neural activity partially correlated over short distances. As self-organization progresses, such preferences are amplified, and groups of neurons develop strong weights to one eye. Figure 2*b* shows the afferent weights of a typical monocular neuron.

The extent of activity correlations on the network determines the size of the monocular neuronal groups. Farther on the map, where the activations are anticorrelated due to lateral inhibition, neurons will develop eye preferences to the opposite eye. As a result, alternating ocular dominance patches develop over the map, as shown in figure 3.[1] In areas between ocular dominance patches, neurons will develop approximately equal strengths to both eyes and become binocular, like the one shown in figure 2*c*.

The width and number of ocular dominance columns in the network (and therefore, the wavelength of ocular dominance) depends on the input correlations (figure 4). When inputs in the two eyes become more correlated ($\rho < 1$), the activations produced by the two inputs in the network overlap closely and activity correlations become shorter range. By Hebbian adaptation, lateral inhibition concentrates in the neighborhood of each neuron, and the distance at which activations becomes anticorrelated decreases. Therefore, smaller monocular patches develop, and the ocular dominance wavelength decreases. Similar dependence was very recently observed in the cat primary visual cortex (Löwel 1994). The LISSOM model demonstrates that the adapting lateral interactions and recurrent activity dynamics regulate the wavelength, and suggests how these processes help the cortex develop feature detectors at a scale

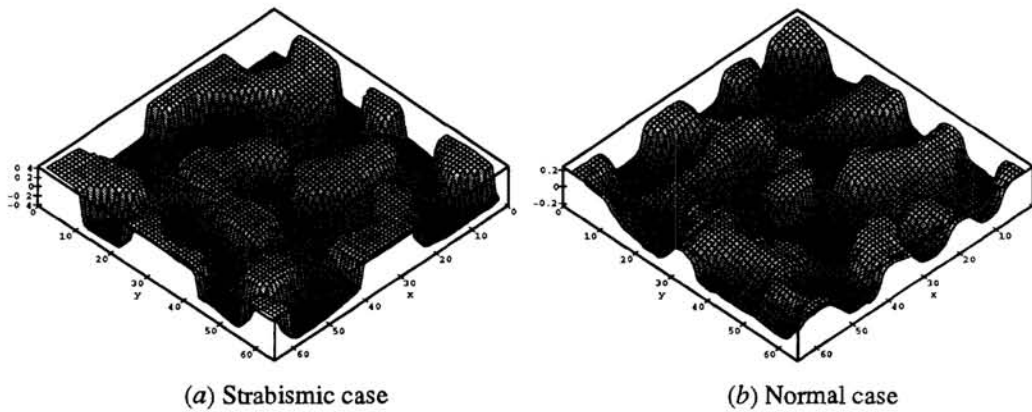

(a) Strabismic case                                    (b) Normal case

Figure 4: **Ocular dominance wavelength in strabismic and normal models**. In the strabismic case, there are no between-eye correlations ($\rho = 1$), and broad ocular dominance columns are produced (a). With normal, partial between-eye correlations ($\rho = 0.45$ in this example), narrower stripes are formed (b). As a result, there are more ocular dominance columns in the normal case and the ocular dominance wavelength is smaller.

that matches the input correlations.

As eye preferences develop, left or right eye input tends to cause activity only in the left or right ocular dominance patches. Activity patterns in areas of the network with the same ocular dominance tend to be highly correlated because they are caused by the same input spot. Therefore, the long-range lateral connections between similar eye preference areas become stronger, and those between opposite areas weaker. After the weak lateral connections are eliminated, the initially wide-ranging connections are pruned, and eventually only connect areas of similar ocular dominance as shown in figure 3. Binocular neurons between ocular dominance patches will see some correlated activity in both the neighboring areas, and maintain connections to both ocular dominance columns (figure 3b).

The lateral connection patterns shown above closely match observations in the primary visual cortex. Löwel and Singer (1992) observed that when between-eye correlations are abolished in kittens by surgically induced strabismus, long-range lateral connections primarily link areas of the same ocular dominance. However, binocular neurons, located between ocular dominance columns, retained connections to both eye regions. The receptive field model confirms that such patterned lateral connections develop based on correlated neuronal activity, and demonstrates that they can self-organize simultaneously with ocular dominance columns. The model also predicts that the long-range connections have an inhibitory function.

## 4   Discussion

In LISSOM, evolving lateral interactions and dynamic activity patterns are explicitly modeled. Therefore, LISSOM has several novel properties that set it apart from other self-organizing models of the cortex.

Previous models (e.g. Goodhill 1993; Miller et al.1989; Obermayer et al.1992; von der Malsburg 1973) have concentrated only on forming ordered topographic maps where clusters of adjacent neurons assume similar response properties such as ocular dominance or orientation preference. The lateral connections in LISSOM, in addition, adapt to encode correlations be-

tween the responses.[2] This property can be potentially very useful in models of cortical function. While afferent connections learn to detect the significant features in the input space (such as ocularity or orientation), the lateral connections can learn correlations between these features (such as Gestalt principles), and thereby form a basis for feature grouping.

As an illustration, consider a single spot of light presented to the left eye. The spot causes disjoint activity patterns in the left-eye-dominant patches. How can these multiple activity patterns be recognized as representing the same spatially coherent entity? As proposed by Singer et al. (1990), the long-range lateral connections between similar ocular dominance columns could synchronize cortical activity, and form a coherently firing assembly of neurons. The spatial coherence of the spot will then be represented by temporal coherence of neural activity. LISSOM can be potentially extended to model such feature binding.

Even after the network has self-organized, the lateral and afferent connections remain plastic and in a continuously-adapting dynamic equilibrium with the input. Therefore, the receptive field properties of neurons can dynamically readapt when the activity correlations in the network are forced to change. For example, when a small area of the cortex is set inactive (or lesioned), the sharply-tuned afferent weight profiles of the neurons surrounding that region expand in size, and neurons begin to respond to the stimuli that previously activated only the lesioned area (Sirosh and Miikkulainen 1994b, 1994c). This expansion of receptive fields is reversible, and when the lesion is repaired, neurons return to their original tuning. Similar changes occur in response to retinal lesions as well. Such dynamic expansions of receptive fields have been observed in the visual cortex (Pettet and Gilbert 1992). The LISSOM model demonstrates that such plasticity is a consequence of the same self-organizing mechanisms that drive the development of cortical maps.

## 5 Conclusion

The LISSOM model shows how a single local and unsupervised self-organizing process can be responsible for the development of both afferent and lateral connection structures in the primary visual cortex. It suggests that this same developmental mechanism also encodes higher-order visual information such as feature correlations into the lateral connections. The model forms a framework for future computational study of cortical reorganization and plasticity, as well as dynamic perceptual processes such as feature grouping and binding.

### Acknowledgments

This research was supported in part by National Science Foundation under grant #IRI-9309273. Computer time for the simulations was provided by the Pittsburgh Supercomputing Center under grants IRI930005P and TRA940029P.

## Footnotes

[1] For a thorough treatment of the mathematical principles underlying the development of ocular dominance columns, see (Goodhill 1993; Miller et al. 1989; von der Malsburg and Singer 1988).

[2] The idea was conceived by von der Malsburg and Singer (1988), but not modeled.

## References

Burkhalter, A., Bernardo, K. L., and Charles, V. (1993). Development of local circuits in human visual cortex. *Journal of Neuroscience*, 13:1916–1931.

Gilbert, C. D., and Wiesel, T. N. (1989). Columnar specificity of intrinsic horizontal and corticocortical connections in cat visual cortex. *Journal of Neuroscience*, 9:2432–2442.

Goodhill, G. (1993). Topography and ocular dominance: a model exploring positive correlations. *Biological Cybernetics*, 69:109–118.

Katz, L. C., and Callaway, E. M. (1992). Development of local circuits in mammalian visual cortex. *Annual Review of Neuroscience*, 15:31–56.

Kohonen, T. (1982). Self-organized formation of topologically correct feature maps. *Biological Cybernetics*, 43:59–69.

Löwel, S. (1994). Ocular dominance column development: Strabismus changes the spacing of adjacent columns in cat visual cortex. *Journal of Neuroscience*, 14(12):7451–7468.

Löwel, S., and Singer, W. (1992). Selection of intrinsic horizontal connections in the visual cortex by correlated neuronal activity. *Science*, 255:209–212.

Malach, R., Amir, Y., Harel, M., and Grinvald, A. (1993). Relationship between intrinsic connections and functional architecture revealed by optical imaging and in vivo targeted biocytin injections in the primate striate cortex. *Proceedings of the National Academy of Sciences, USA*, 90:10469–10473.

Miller, K. D. (1994). A model for the development of simple cell receptive fields and the ordered arrangement of orientation columns through activity-dependent competition between on- and off-center inputs. *Journal of Neuroscience*, 14:409–441.

Miller, K. D., Keller, J. B., and Stryker, M. P. (1989). Ocular dominance column development: Analysis and simulation. *Science*, 245:605–615.

Obermayer, K., Blasdel, G. G., and Schulten, K. J. (1992). Statistical-mechanical analysis of self-organization and pattern formation during the development of visual maps. *Physical Review A*, 45:7568–7589.

Pettet, M. W., and Gilbert, C. D. (1992). Dynamic changes in receptive-field size in cat primary visual cortex. *Proceedings of the National Academy of Sciences, USA*, 89:8366–8370.

Singer, W., Gray, C., Engel, A., König, P., Artola, A., and Bröcher, S. (1990). Formation of cortical cell assemblies. In *Cold Spring Harbor Symposia on Quantitative Biology, Vol. LV*, 939–952. Cold Spring Harbor, NY: Cold Spring Harbor Laboratory.

Sirosh, J., and Miikkulainen, R. (1993). How lateral interaction develops in a self-organizing feature map. In *Proceedings of the IEEE International Conference on Neural Networks (San Francisco, CA)*, 1360–1365. Piscataway, NJ: IEEE.

Sirosh, J., and Miikkulainen, R. (1994a). Cooperative self-organization of afferent and lateral connections in cortical maps. *Biological Cybernetics*, 71(1):66–78.

Sirosh, J., and Miikkulainen, R. (1994b). Modeling cortical plasticity based on adapting lateral interaction. In *The Neurobiology of Computation: Proceedings of the Annual Computational Neuroscience Meeting*. Dordrecht; Boston: Kluwer. In Press.

Sirosh, J., and Miikkulainen, R. (1994c). A neural network model of topographic reorganization following cortical lesions. In *Proceedings of the World Congress on Computational Medicine, Public Health and Biotechnology* (Austin, TX). World Scientific. In Press.

von der Malsburg, C. (1973). Self-organization of orientation-sensitive cells in the striate cortex. *Kybernetik*, 15:85–100.

von der Malsburg, C., and Singer, W. (1988). Principles of cortical network organization. In Rakic, P., and Singer, W., editors, *Neurobiology of Neocortex*, 69–99. New York: Wiley.